# A Sampled Texture Prior for Image Super-Resolution

**Lyndsey C. Pickup, Stephen J. Roberts and Andrew Zisserman**
Robotics Research Group
Department of Engineering Science
University of Oxford
Parks Road, Oxford, OX1 3PJ
{elle,sjrob,az}@robots.ox.ac.uk

## Abstract

Super-resolution aims to produce a high-resolution image from a set of one or more low-resolution images by recovering or inventing plausible high-frequency image content. Typical approaches try to reconstruct a high-resolution image using the sub-pixel displacements of several low-resolution images, usually regularized by a generic smoothness prior over the high-resolution image space. Other methods use training data to learn low-to-high-resolution matches, and have been highly successful even in the single-input-image case. Here we present a domain-specific image prior in the form of a p.d.f. based upon sampled images, and show that for certain types of super-resolution problems, this sample-based prior gives a significant improvement over other common multiple-image super-resolution techniques.

## 1 Introduction

The aim of super-resolution is to take a set of one or more low-resolution input images of a scene, and estimate a higher-resolution image. If there are several low resolution images available with sub-pixel displacements, then the high frequency information of the super-resolution image can be increased.

In the limiting case when the input set is just a single image, it is impossible to recover any high-frequency information faithfully, but much success has been achieved by training models to learn patchwise correspondences between low-resolution and possible high-resolution information, and stitching patches together to form the super-resolution image [1]. A second approach uses an unsupervised technique where latent variables are introduced to model the mean intensity of groups of surrounding pixels [2].

In cases where the high-frequency detail is recovered from image displacements, the models tend to assume that each low-resolution image is a subsample from a true high-resolution image or continuous scene. The generation of the low-resolution inputs can then be expressed as a degradation of the super-resolution image, usually by applying an image homography, convolving with blurring functions, and subsampling [3, 4, 5, 6, 7, 8, 9].

Unfortunately, the ML (*maximum likelihood*) super-resolution images obtained by revers-

ing the generative process above tend to be poorly conditioned and susceptible to high-frequency noise. Most approaches to multiple-image super-resolution use a MAP (*maximum a-posteriori*) approach to regularize the solution using a prior distribution over the high-resolution space. Gaussian process priors [4], Gaussian MRFs (Markov Random Fields) and Huber MRFs [3] have all been proposed as suitable candidates.

In this paper, we consider an image prior based upon samples taken from other images, inspired by the use of non-parametric sampling methods in texture synthesis [10]. This texture synthesis method outperformed many other complex parametric models for texture representation, and produces perceptively correct-looking areas of texture given a sample texture seed. It works by finding texture patches similar to the area around a pixel of interest, and estimating the intensity of the central pixel from a histogram built up from similar samples. We turn this approach around to produce an image prior by finding areas in our sample set that are similar to patches in our super-resolution image, and evaluate how well they match, building up a p.d.f. over the high-resolution image. In short, given a set of low resolution images and example images of textures in the same class at the higher resolution, our objective is to construct a super-resolution image using a prior that is sampled from the example images.

Our method differs from the previous super-resolution methods of [1, 7] in two ways: first, we use our training images to estimate a distribution rather than learn a discrete set of low-resolution to high-resolution matches from which we must build up our output image; second, since we are using more than one image, we naturally fold in the extra high-frequency information available from the low-resolution image displacements.

We develop our model in section 2, and expand upon some of the implementation details in section 3, as well as introducing the Huber prior model against which most of the comparisons in this paper are made. In section 4 we display results obtained with our method on some simple images, and in section 5 we discuss these results and future improvements.

## 2 The model

In this section we develop the mathematical basis for our model. The main contribution of this work is in the construction of the prior over the super-resolution image, but first we will consider the generative model for the low-resolution image generation, which closely follows the approaches of [3] and [4]. We have $K$ low-resolution images $\boldsymbol{y}^{(k)}$, which we assume are generated from the super-resolution image $\boldsymbol{x}$ by

$$\boldsymbol{y}^{(k)} = \boldsymbol{W}^{(k)}\boldsymbol{x} + \boldsymbol{\epsilon_G}^{(k)} \tag{1}$$

where $\boldsymbol{\epsilon_G}$ is a vector of *i.i.d.* Gaussians $\epsilon_G \sim \mathcal{N}(0, \beta_G^{-1})$, and $\beta_G$ is the noise precision. The construction of $\boldsymbol{W}$ involves mapping each low-resolution pixel into the space of the super-resolution image, and performing a convolution with a point spread function. The constructions given in [3] and [4] are very similar, though the former uses bilinear interpolation to achieve a more accurate approximation.

We begin by assuming that the image registration parameters may be determined *a priori*, so each input image has a corresponding set of registration parameters $\boldsymbol{\theta}^{(k)}$. We may now construct the likelihood function

$$p(\boldsymbol{y}^{(k)}|\boldsymbol{x}, \boldsymbol{\theta}^{(k)}) = \left(\frac{\beta_G}{2\pi}\right)^{M/2} \exp\left[-\frac{\beta_G}{2}||\boldsymbol{y}^{(k)} - \boldsymbol{W}^{(k)}\boldsymbol{x}||^2\right] \tag{2}$$

where each input image is assumed to have $M$ pixels (and the super-resolution image $N$ pixels).

The ML solution for $\boldsymbol{x}$ can be found simply by maximizing equation 2 with respect to $\boldsymbol{x}$,

which is equivalent to minimizing the negative log likelihood

$$-\log p(\{\boldsymbol{y}^{(k)}\}|\boldsymbol{x},\{\boldsymbol{\theta}^{(k)}\}) \propto \sum_{k=1}^{K} ||\boldsymbol{y}^{(k)} - \boldsymbol{W}^{(k)}\boldsymbol{x}||^2, \tag{3}$$

though super-resolved images recovered in this way tend to be dominated by a great deal of high-frequency noise.

To address this problem, a prior over the super-resolution image is often used. In [4], the authors restricted themselves to Gaussian process priors, which made their estimation of the registration parameters $\boldsymbol{\theta}$ tractable, but encouraged smoothness across $\boldsymbol{x}$ without any special treatment to allow for edges. The Huber Prior was used successfully in [3] to penalize image gradients while being less harsh on large image discontinuities than a Gaussian prior. Details of the Huber prior are given in section 3.

If we assume a uniform prior over the input images, the posterior distribution over $\boldsymbol{x}$ is of the form

$$p(\boldsymbol{x}|\{\boldsymbol{y}^{(k)}, \boldsymbol{\theta}^{(k)}\}) \quad \propto \quad p(\boldsymbol{x}) \prod_{k=1}^{K} p(\boldsymbol{y}^{(k)}|\boldsymbol{x}, \boldsymbol{\theta}^{(k)}). \tag{4}$$

To build our expression for $p(\boldsymbol{x})$, we adopt the philosophy of [10], and sample from other example images rather than developing a parametric model. A similar philosophy was used in [11] for image-based rendering. Given a small image patch around any particular pixel, we can learn a distribution for the central pixel's intensity value by examining the values at the centres of similar patches from other images. Each pixel $x_i$ has a neighbourhood region $\mathcal{R}(x_i)$ consisting of the pixels around it, but not including $x_i$ itself. For each $\mathcal{R}(x_i)$, we find the closest neighbourhood patch in the set of sampled patches, and find the central pixel associated with this nearest neighbour, $L_{\mathcal{R}}(x_i)$. The intensity of our original pixel is then assumed to be Gaussian distributed with mean equal to the intensity of this central pixel, and with some precision $\beta_T$,

$$x_i \sim \mathcal{N}(L_{\mathcal{R}}(x_i), \beta_T^{-1}) \tag{5}$$

leading us to a prior of the form

$$p(\boldsymbol{x}) = \left(\frac{\beta_T}{2\pi}\right)^{N/2} \exp\left[-\frac{\beta_T}{2}||\boldsymbol{x} - L_{\mathcal{R}}(\boldsymbol{x})||^2\right]. \tag{6}$$

Inserting this prior into equation 4, the posterior over $\boldsymbol{x}$, and taking the negative log, we have

$$-\log p(\boldsymbol{x}|\{\boldsymbol{y}^{(k)}, \boldsymbol{\theta}^{(k)}\}) \quad \propto \quad \beta||\boldsymbol{x} - L_{\mathcal{R}}(\boldsymbol{x})||^2 + \sum_{k=1}^{K} ||\boldsymbol{y}^{(k)} - \boldsymbol{W}^{(k)}\boldsymbol{x}||^2 + c, \tag{7}$$

where the right-hand side has been scaled to leave a single unknown ratio $\beta$ between the data error term and the prior term, and includes an arbitrary constant $c$. Our super-resolution image is then just $\arg\min_{\boldsymbol{x}}(\mathcal{L})$, where

$$\mathcal{L} = \beta||\boldsymbol{x} - L_{\mathcal{R}}(\boldsymbol{x})||^2 + \sum_{k=1}^{K} ||\boldsymbol{y}^{(k)} - \boldsymbol{W}^{(k)}\boldsymbol{x}||^2. \tag{8}$$

## 3 Implementation details

We optimize the objective function of equation 8 using scaled conjugate gradients (SCG) to obtain an approximation to our super-resolution image. This requires an expression for

the gradient of the function with respect to $\boldsymbol{x}$. For speed, we approximate this by

$$\frac{d\mathcal{L}}{d\boldsymbol{x}} = 2\beta\big(\boldsymbol{x} - L_{\mathcal{R}}(\boldsymbol{x})\big) - \frac{2}{K}\sum_{k=1}^{K}\boldsymbol{W}^{(k)T}\big(\boldsymbol{y}^{(k)} - \boldsymbol{W}^{(k)}\boldsymbol{x}\big), \tag{9}$$

which assumes that small perturbations in the neighbours of $\boldsymbol{x}$ will not change the value returned by $L_{\mathcal{R}}(\boldsymbol{x})$. This is obviously not necessarily the case, but leads to a more efficient algorithm. The same k-nearest-neighbour variation introduced in [10] could be adopted to smooth this response.

Our image patch regions $\mathcal{R}(x_i)$ are square windows centred on $x_i$, and pixels near the edge of the image are supported using the average image of [3] extending beyond the edge of the super-resolution image. To compute the nearest region in the example images, patches are normalized to sum to unity, and centre weighted as in [10] by a 2-dimensional Gaussian. The width of the image patches used, and of the Gaussian weights, depends very much upon the scales of the textures present in the image. Our images intensities were in the range $[0, 1]$, and all the work so far has been with grey-scale images.

Most of our results with this sample-based prior are compared to super-resolution images obtained using the Huber prior used in [3]. Other edge-preserving functions are discussed in [12], though the Huber function performed better than these as a prior in this case. The Huber potential function is given by

$$\rho(x) = \left\{ \begin{array}{ll} x^2, & \text{if } |x| \leq \alpha \\ 2\alpha|x| - \alpha^2, & \text{otherwise.} \end{array} \right. \tag{10}$$

If $\boldsymbol{G}$ is a matrix which pre-multiplies $\boldsymbol{x}$ to give a vector of first-order approximations to the magnitude of the image gradient in the horizontal, vertical, and two diagonal directions, then the Huber prior we use is of the form:

$$p(\boldsymbol{x}) = \frac{1}{Z}\exp\left[-\gamma\sum_{i=1}^{4N}\rho((\boldsymbol{Gx})_i)\right] \tag{11}$$

for some prior strength $\gamma$, $Z$ is the partition function, and $(\boldsymbol{Gx})$ is the $4N \times 1$ column vector of approximate derivatives of $\boldsymbol{x}$ in the four directions mentioned above.

Plugging this into the posterior distribution of equation 4 leads to a Huber MAP image $\boldsymbol{x}_H$ which minimizes the negative log probability

$$\mathcal{L}_H = \beta\sum_{i=1}^{4N}\rho((\boldsymbol{Gx})_i) + \sum_{k=1}^{K}||\boldsymbol{y}^{(k)} - \boldsymbol{W}^{(k)}\boldsymbol{x}||^2, \tag{12}$$

where again the r.h.s. has been scaled so that $\beta$ is the single unknown ratio parameter. We also optimize this by SCG, using the full analytic expression for $\frac{d\mathcal{L}_H}{d\boldsymbol{x}}$.

## 4 Preliminary results

To test the performance of our texture-based prior, and compare it with that of the Huber prior, we produced sets of input images by running the generative model of equation 1 in the forward direction, introducing sub-pixel shifts in the $x$- and $y$-directions, and a small rotation about the viewing axis. We added varying amounts of Gaussian noise ($2/256$, $6/256$ and $12/256$, grey levels) and took varying number of these images ($2, 5, 10$) to produce nine separate sets of low-resolution inputs from each of our initial "ground-truth" high resolution images. Figure 1 shows three $100 \times 100$ pixel ground truth images, each accompanied by corresponding $40 \times 40$ pixel low-resolution images generated from the

ground truth images at half the resolution, with $6/256$ levels of noise. Our aim was to reconstruct the central $50 \times 50$ pixel section of the original ground truth image. Figure 2 shows the example images from which our texture samples patches were taken [1] – note that these do not overlap with the sections used to generate the low-resolution images.

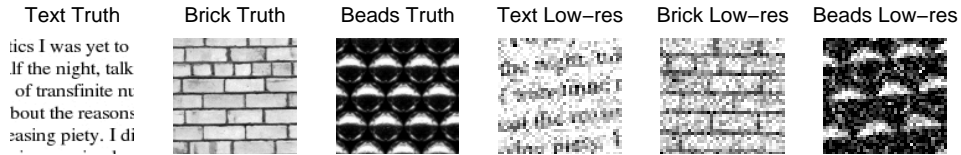

Figure 1: Left to right: ground truth text, ground truth brick, ground truth beads, low-res text, low-res brick and low-res beads.

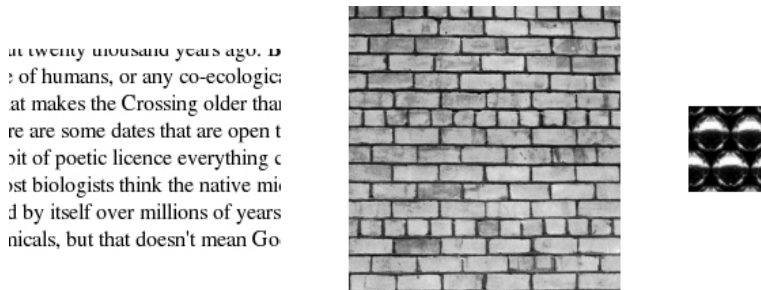

Figure 2: Left: Text sample ($150 \times 200$ pixels). Centre: Brick sample ($200 \times 200$ pixels). Right: Beads sample ($60 \times 60$ pixels).

Figure 3 shows the difference in super-resolution image quality that can be obtained using the sample-based prior over the Huber prior using identical input sets as described above.

For each Huber super-resolution image, we ran a set of reconstructions, varying the Huber parameter $\alpha$ and the prior strength parameter $\beta$. The image shown for each input number/noise level pair is the one which gave the minimum RMS error when compared to the ground-truth image; these are very close to the "best" images chosen from the same sets by a human subject.

The images shown for the sample-based prior are again the best (in the sense of having minimal RMS error) of several runs per image. We varied the size of the sample patches from 5 to 13 pixels in edge length – computational cost meant that larger patches were not considered. Compared to the Huber images, we tried relatively few different patch size and $\beta$-value combinations for our sample-based prior; again, this was due to our method taking longer to execute than the Huber method. Consequently, the Huber parameters are more likely to lie close to their own optimal values than our sample-based prior parameters are.

We also present images recovered using a "wrong" texture. We generated ten low-resolution images from a picture of a leaf, and used texture samples from a small black-and-white spiral in our reconstruction (Figure 4). A selection of results are shown in Figure 5, where we varied the $\beta$ parameter governing the prior's contribution to the output image.

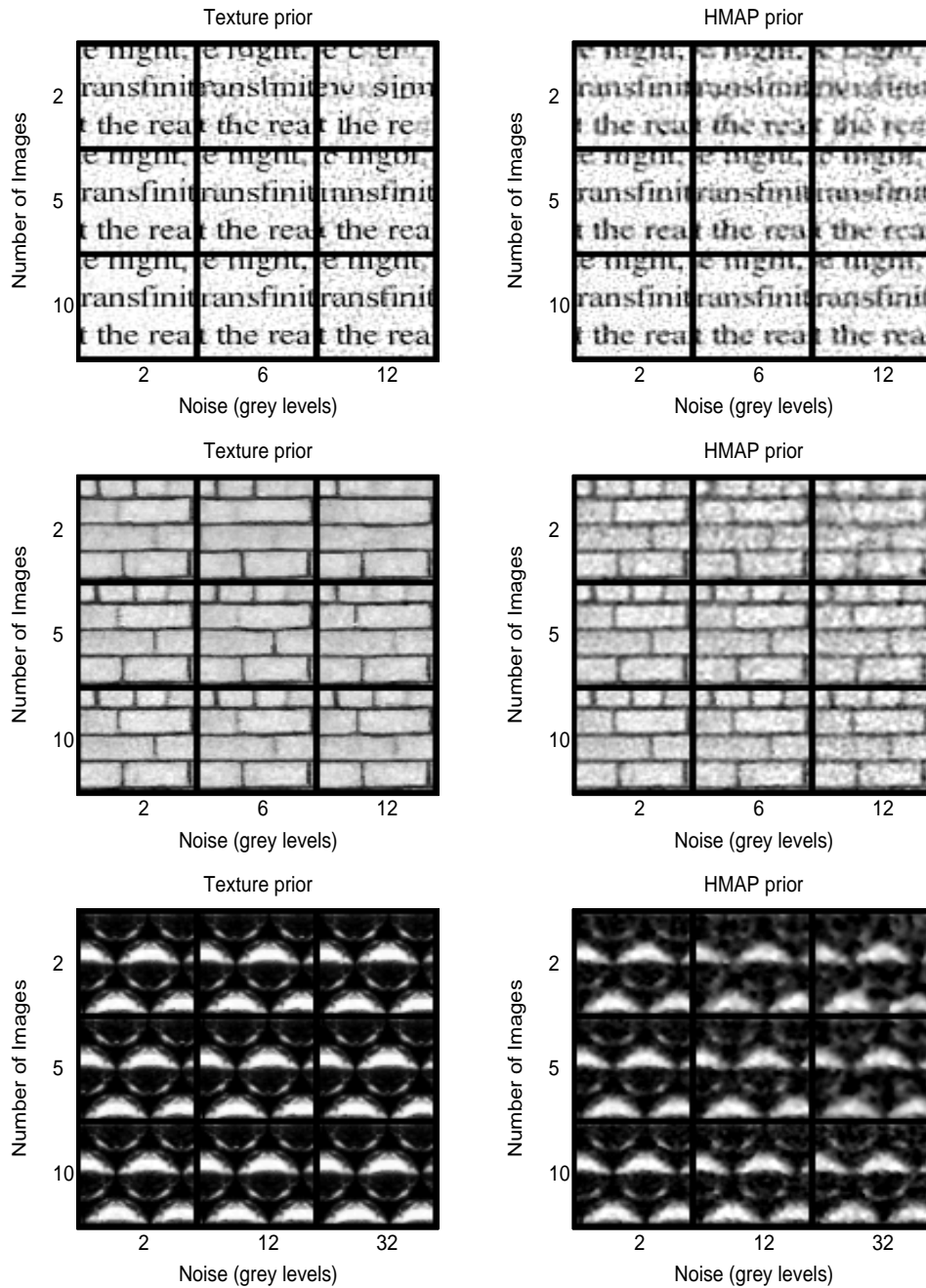

Figure 3: Recovering the super-resolution images at a zoom factor of 2, using the texture-based prior (left column of plots) and the Huber MRF prior (right column of plots). The text and brick datasets contained 2, 6, 12 grey levels of noise, while the beads dataset used 2, 12 and 32 grey levels. Each image shown is the best of several attempts with varying prior strengths, Huber parameter (for the Huber MRF prior images) and patch neighbourhood sizes (for the texture-based prior images).

Using a low value gives an image not dissimilar to the ML solution; using a significantly higher value makes the output follow the form of the prior much more closely, and here this means that the grey values get lost as the evidence for them from the data term is swamped by the black-and-white pattern of the prior.

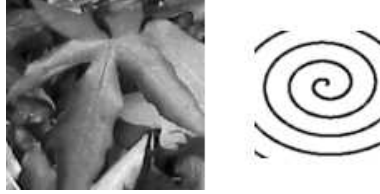

Figure 4: The original $120 \times 120$ high-resolution image (left), and the $80 \times 80$ pixel "wrong" texture sample image (right).

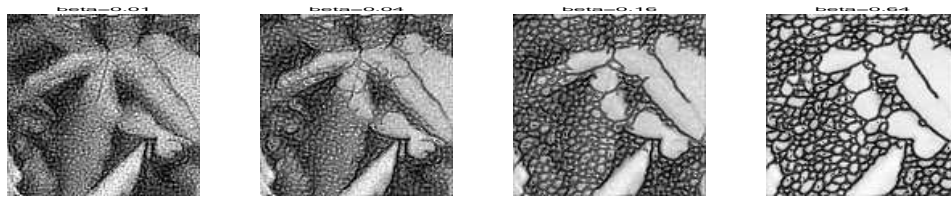

Figure 5: Four $120 \times 120$ super-resolution images are shown on the lower row, reconstructed using different values of the prior strength parameter $\beta$: 0.01, 0.04, 0.16, 0.64, from left to right.

## 5 Discussion and further considerations

The images of Figure 3 show that our prior offers a qualitative improvement over the generic prior, especially when few input images are available.

Quantitively, our method gives an RMS error of approximately 25 grey levels from only 2 input images with 2 grey levels of additive Gaussian noise on the text input images, whereas the best Huber prior super-resolution image for that image set and noise level uses all 10 available input images, and still has an RMS error score of almost 30 grey levels.

Figure 6 plots the RMS errors from the Huber and sample-based priors against each other. In all cases, the sample-based method fares better, with the difference most notable in the text example.

In general, larger patch sizes ($11 \times 11$ pixels) give smaller errors for the noisy inputs, while small patches ($5 \times 5$) are better for the less noisy images. Computational costs mean we limited the patch size to no more than $13 \times 13$, and terminated the SCG optimization algorithm after approximately 20 iterations.

In addition to improving the computational complexity of our algorithm implementation, we can extend this work in several directions. Since in general the textures for the prior will not be invariant to rotation and scaling, consideration of the registration of the input images will be necessary. The optimal patch size will be a function of the image textures, so learning this as a parameter of an extended model, in a similar way to how [4] learns the point-spread function for a set of input images, is another direction of interest.

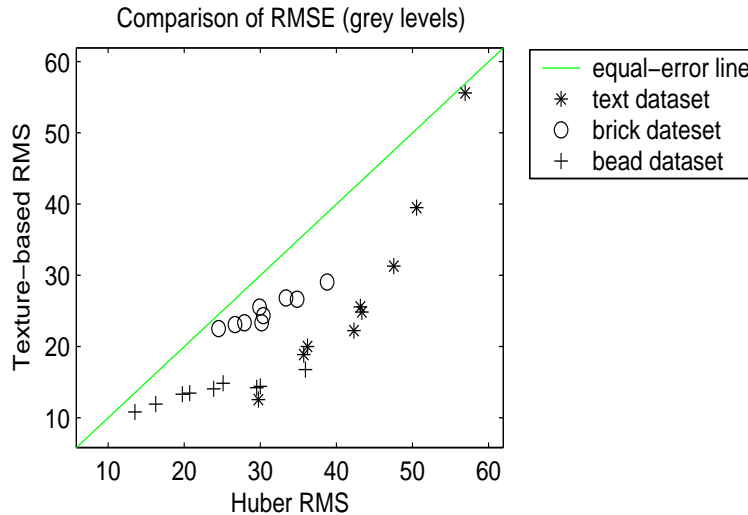

Figure 6: Comparison of RMS errors in reconstructing the text, brick and bead images using the Huber and sample-based priors.

## Footnotes

[1]Text grabbed from Greg Egan's novella *Oceanic*, published online at the author's website. Brick image from the Brodatz texture set. Beads image from *http://textures.forrest.cz/*.

## References

[1] W. T. Freeman, T. R. Jones, and E. C. Pasztor. Example-based super-resolution. *IEEE Computer Graphics and Applications*, 22(2):56–65, March/April 2002.

[2] A. J. Storkey. Dynamic structure super-resolution. In S. Thrun S. Becker and K. Obermayer, editors, *Advances in Neural Information Processing Systems 15*, pages 1295–1302. MIT Press, Cambridge, MA, 2003.

[3] D. P. Capel. *Image Mosaicing and Super-resolution*. PhD thesis, University of Oxford, 2001.

[4] M. E. Tipping and C. M. Bishop. Bayesian image super-resolution. In S. Thrun S. Becker and K. Obermayer, editors, *Advances in Neural Information Processing Systems 15*, pages 1279–1286. MIT Press, Cambridge, MA, 2003.

[5] M. Irani and S. Peleg. Improving resolution by image registration. *CVGIP: Graphical Models and Image Processing*, 53:231–239, 1991.

[6] M. Irani and S. Peleg. Motion analysis for image enhancement:resolution, occlusion, and transparency. *Journal of Visual Communication and Image Representation*, 4:324–335, 1993.

[7] S. Baker and T. Kanade. Limits on super-resolution and how to break them. *IEEE Transactions on Pattern Analysis and Machine Intelligence*, 24(9):1167–1183, 2002.

[8] R. R. Schultz and R. L. Stevenson. Extraction of high-resolution frames from video sequences. *IEEE Transactions on Image Processing*, 5(6):996–1011, June 1996.

[9] P. Cheeseman, B. Kanefsky, R. Kraft, J. Stutz, and B. Hanson. Super-resolved surface reconstruction from multiple images. In Glenn R. Heidbreder, editor, *Maximum Entropy and Bayesian Methods*, pages 293–308. Kluwer Academic Publishers, Dordrecht, the Netherlands, 1996.

[10] A. A. Efros and T. K. Leung. Texture synthesis by non-parametric sampling. In *IEEE International Conference on Computer Vision*, pages 1033–1038, Corfu, Greece, September 1999.

[11] A. Fitzgibbon, Y. Wexler, and A. Zisserman. Image-based rendering using image-based priors. In *Proceedings of the International Conference on Computer Vision*, October 2003.

[12] M. J. Black, G. Sapiro, D. Marimont, and D. Heeger. Robust anisotropic diffusion. *IEEE Trans. on Image Processing*, 7(3):421–432, 1998.
